# VLSI Model of Primate Visual Smooth Pursuit

**Ralph Etienne-Cummings**
Department of Electrical Engineering,
Southern Illinois University, Carbondale,
IL 62901

**Jan Van der Spiegel**
Moore School of Electrical Engineering,
University of Pennsylvania, Philadelphia,
PA 19104

**Paul Mueller**
Corticon, Incorporated,
3624 Market Str, Philadelphia,
PA 19104

## Abstract

A one dimensional model of primate smooth pursuit mechanism has been implemented in 2 $\mu$m CMOS VLSI. The model consolidates Robinson's negative feedback model with Wyatt and Pola's positive feedback scheme, to produce a smooth pursuit system which zero's the velocity of a target on the retina. Furthermore, the system uses the current eye motion as a predictor for future target motion. Analysis, stability and biological correspondence of the system are discussed. For implementation at the focal plane, a local correlation based visual motion detection technique is used. Velocity measurements, ranging over 4 orders of magnitude with < 15% variation, provides the input to the smooth pursuit system. The system performed successful velocity tracking for high contrast scenes. Circuit design and performance of the complete smooth pursuit system is presented.

## 1 INTRODUCTION

The smooth pursuit mechanism of primate visual systems is vital for stabilizing a region of the visual field on the retina. The ability to stabilize the image of the world on the retina has profound architectural and computational consequences on the retina and visual cortex, such as reducing the required size, computational speed and communication hardware and bandwidth of the visual system (Bandera, 1990; Eckert and Buchsbaum, 1993). To obtain similar benefits in active machine vision, primate smooth pursuit can be a powerful model for gaze control. The mechanism for smooth pursuit in primates was initially believed to be composed of a simple negative feedback system which attempts to zero the motion of targets on the fovea, figure 1(a) (Robinson, 1965). However, this scheme does not account for many psychophysical properties of smooth

pursuit, which led Wyatt and Pola (1979) to proposed figure 1(b), where the eye movement signal is added to the target motion in a positive feed back loop. This mechanism results from their observation that eye motion or apparent target motion increases the magnitude of pursuit motion even when retinal motion is zero or constant. Their scheme also exhibited predictive qualities, as reported by Steinbach (1976). The smooth pursuit model presented in this paper attempts the consolidate the two models into a single system which explains the findings of both approaches.

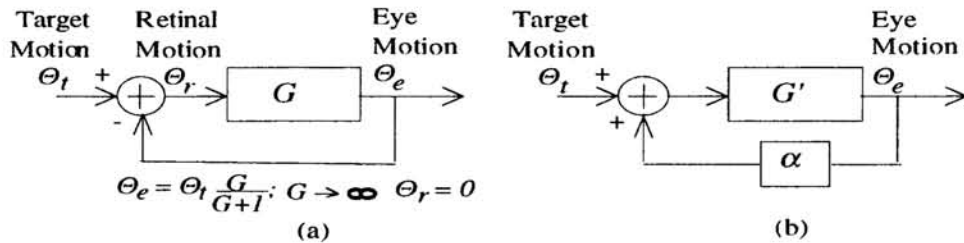

Figure 1: System Diagrams of Primate Smooth Pursuit Mechanism. (a) Negative feedback model by Robinson (1965). (b) Positive feedback model by Wyatt and Pola (1979).

The velocity based smooth pursuit implemented here attempts to zero the relative velocity of the retina and target. The measured retinal velocity, is zeroed by using positive feedback to accumulate relative velocity error between the target and the retina, where the accumulated value is the current eye velocity. Hence, this model uses the Robinson approach to match target motion, and the Wyatt and Pola positive feed back loop to achieve matching and to predict the future velocity of the target. Figure 2 shows the system diagram of the velocity based smooth pursuit system. This system is analyzed and the stability criterion is derived. Possible computational blocks for the elements in figure 1(b) are also discussed. Furthermore, since this entire scheme is implemented on a single 2 µm CMOS chip, the method for motion detection, the complete tracking circuits and the measured results are presented.

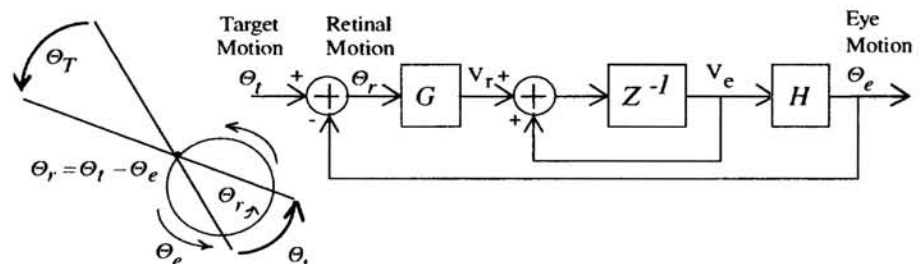

Figure 2: System Diagram of VLSI Smooth Pursuit Mechanism. $\Theta_T$ is target velocity in space, $\Theta_t$ is projected target velocity, $\Theta_e$ is the eye velocity and $\Theta_r$ is the measured retinal velocity.

## 2 VELOCITY BASED SMOOTH PURSUIT

Although figure 1(b) does not indicate how retinal motion is used in smooth pursuit, it provides the only measurement of the projected target motion. The very process of calculating retinal motion realizes negative feed back between the eye movement and the target motion, since retinal motion is the difference between project target and eye motion. If Robinson's model is followed, then the eye movement is simply the amplified version of the retinal motion. If the target disappears from the retina, the eye motion would be zero. However, Steinbach showed that eye movement does not cease when the target fades off and on, indicating that memory is used to predict target motion. Wyatt and Palo showed a direct additive influence of eye movement on pursuit. However, the computational blocks $G'$ and $\alpha$ of their model are left unfilled.

In figure 2, the gain $G$ models the internal gain of the motion detection system, and the internal representation of retinal velocity is then $V_r$. Under zero-slip tracking, the retinal velocity is zero. This is obtained by using positive feed back to correct the velocity error between target, $\Theta_t$, and eye, $\Theta_e$. The delay element represents a memory of the last eye velocity while the current retinal motion is measured. If the target disappears, the eye motion continues with the last value, as recorded by Steinbach, thus anticipating the position of the target in space. The memory also stores the current eye velocity during perfect pursuit. The internal representation of eye velocity, $V_e$, is subsequently amplified by $H$ and used to drive the eye muscles. The impulse response of the system is given in equations (1). Hence, the relationship between eye velocity and target velocity is recursive and given by equations (2). To prove the stability of this system, the retinal velocity can be expressed in terms of the target motion as given in equations (3a). The ideal condition for accurate performance is for GH = 1. However, in practice, gains of different amplifiers

$$\frac{\theta_e}{\theta_r}(z) = GH\frac{z^{-1}}{1-z^{-1}} \ (a); \quad \frac{\theta_e}{\theta_r}(n) = GH[-\delta(n)+u(n)] \ (b) \tag{1}$$

$$\theta_e(n) = \theta_t(n) - \theta_r(n) = GH[-\delta(n)+u(n)] * \theta_r(n) = GH\sum_{k=0}^{n-1}\theta_r(k) \tag{2}$$

$$\theta_r(n) = \theta_t(n)(1-GH)^n \Rightarrow \theta_r(n) = 0 \ if \ GH = 1 \Rightarrow \theta_e(n) = \theta_t(n) \quad (a)$$

$$\theta_r(n) \xrightarrow{n \to \infty} 0 \ if \ |1-GH| < 1 \Rightarrow 0 < GH < 2 \ for \ stability \quad (b) \tag{3}$$

are rarely perfectly matched. Equations (3b) shows that stability is assured for 0<GH< 2. Figure 3 shows a plot of eye motion versus updates for various choices of GH. At each update, the retinal motion is computed. Figure 3(a) shows the eye's motion at the on-set of smooth pursuit. For GH = 1, the eye movement tracks the target's motion exactly, and lags slightly only when the target accelerates. On the other hand, if GH << 1, the eye's motion always lags the target's. If GH -> 2, the system becomes increasing unstable, but converges for GH < 2. The three cases presented correspond to the smooth pursuit system being critically, over and under damped, respectively.

# 3 HARDWARE IMPLEMENTATION

Using the smooth pursuit mechanism described, a single chip one dimensional tracking system has been implemented. The chip has a multi-layered computational architecture, similar to the primate's visual system. Phototransduction, logarithmic compression, edge detection, motion detection and smooth pursuit control has been integrated at the focal-plane. The computational layers can be partitioned into three blocks, where each block is based on a segment of biological oculomotor systems.

## 3.1 IMAGING AND PREPROCESSING

The first three layers of the system mimics the photoreceptors, horizontal cells and bipolar cells of biological retinas. Similar to previous implementations of silicon retinas, the chip uses parasitic bipolar transistors as the photoreceptors. The dynamic range of photoreceptor current is compressed with a logarithmic response in low light and square root response in bright light. The range compress circuit represents 5-6 orders of magnitude of light intensity with 3 orders of magnitude of output current dynamic range. Subsequently, a passive resistive network is used to realize a discrete implementation of a Laplacian edge detector. Similar to the rods and cones system in primate retinas, the response time, hence the maximum detectable target speed, is ambient intensity dependent (160 (12.5) µs in 2.5 (250) µW/cm$^2$). However, this does prevent the system from handling fast targets even in dim ambient lighting.

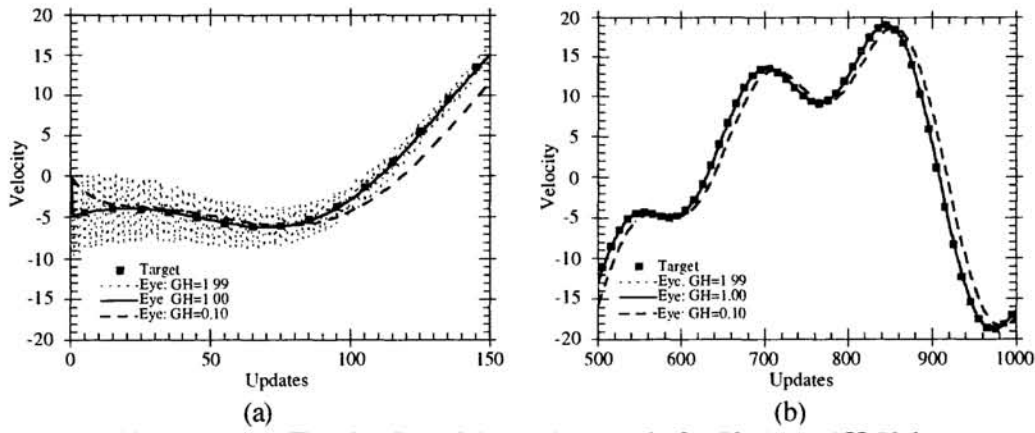

Figure 3: (a) The On-Set of Smooth Pursuit for Various GH Values.
(b) Steady-State Smooth Pursuit.

## 3.2  MOTION MEASUREMENT

This computational layer measures retinal motion. The motion detection technique implemented here differs from those believed to exist in areas V1 and MT of the primate visual cortex. Alternatively, it resembles the fly's and rabbit's retinal motion detection system (Reichardt, 1961; Barlow and Levick, 1965; Delbruck, 1993). This is not coincidental, since efficient motion detection at the focal plane must be performed in a small areas and using simple computational elements in both systems.

The motion detection scheme is a combination of local correlation for direction determination, and pixel transfer time measurement for speed. In this framework, motion is defined as the disappearance of an object, represented as the zero-crossings of its edges, at a pixel, followed by its re-appearance at a neighboring pixel. The (dis)appearance of the zero-crossing is determined using the (negative) positive temporal derivative at the pixel. Hence, motion is detected by AND gating the positive derivative of the zero-crossing of the edge at one pixel with the negative derivative at a neighboring pixel. The direction of motion is given by the neighboring pixel from which the edge disappeared. Provided that motion has been detected at a pixel, the transfer time of the edge over the pixel's finite geometry is inversely proportional to its speed.

Equation (4) gives the mathematical representation of the motion detection process for an object moving in +x direction. In the equation, $f_t(I{:}k,y,t)$ is the temporal response of pixel $k$ as the zero crossing of an edge of an object passes over its $2a$ aperture. Equation (4) gives the direction of motion, while equation (5) gives the speed. The schematic of

$$motion_{-x} = [\tfrac{\partial}{\partial t}f_t(I{:}k,y,t) > 0 ][\tfrac{\partial}{\partial t}f_t(I{:}k+1,y,t) < 0 ]=0 \qquad (a)$$

$$motion_{+x} = [\tfrac{\partial}{\partial t}f_t(I{:}k-1,y,t) < 0 ][\tfrac{\partial}{\partial t}f_t(I{:}k,y,t) > 0 ] \qquad (b) \qquad (4)$$

$$= \delta[t - \frac{2a(k-n)-a}{v_x}]\delta[x-2ak]$$

$$Motion : t_m = \frac{2a(k-n)-a}{v_x}; \quad Disappear : t_d = \frac{2a(k-n)+a}{v_x}$$

$$Speed_{+x} = \frac{1}{t_d - t_m} = \frac{v_x}{2a} \qquad (5)$$

the VLSI circuit of the motion detection model is shown in figure 4(a). Figure 4(b) shows reciprocal of the measured motion pulse-width for 1 D motion. The on-chip speed, $\Theta_t$, is the projected target speed. The measured pulse-widths span 3-4 orders magnitude,

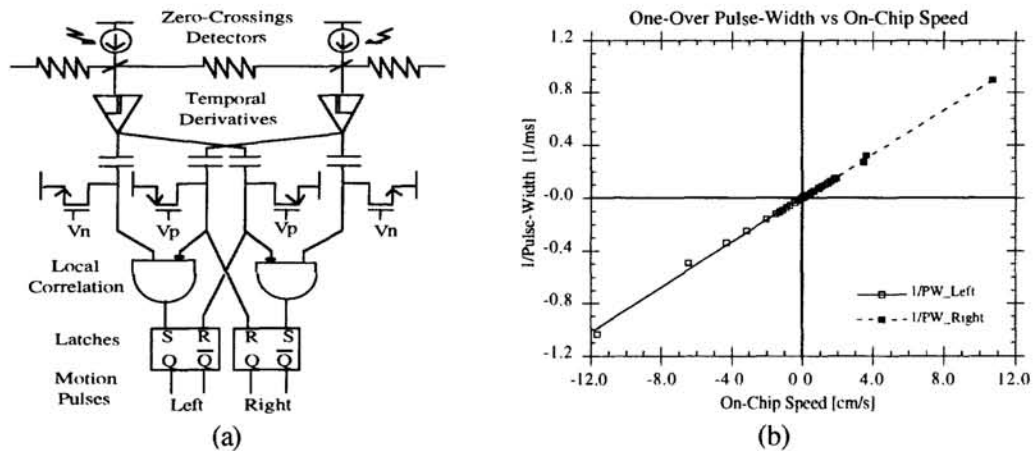

Figure 4: (a) Schematic of the Motion Detection Circuit. (b) Measured Output of the Motion Detection Circuit.

depending on the ambient lighting, and show less than 15% variation between chips, pixels, and directions (Etienne-Cummings, 1993).

## 3.3 THE SMOOTH PURSUIT CONTROL SYSTEM

The one dimensional smooth pursuit system is implemented using a 9 x 1 array of motion detectors. Figure 5 shows the organization of the smooth pursuit chip. In this system, only diverging motion is computed to reduce the size of each pixel. The outputs of the motion detectors are grouped into one global motion signal per direction. This grouping is performed with a simple, but delayed, OR, which prevents pulses from neighboring motion cells from overlapping. The motion pulse trains for each direction are XOR gated, which allows a single integrator to be used for both directions, thus limiting mis-match. The final value of the integrator is inversely proportional to the target's speed. The OR gates conserve the direction of motion. The reciprocal of the integrator voltage is next computed using the linear mode operation of a MOS transistor (Etienne-Cummings, 1993). The unipolar integrated pulse allows a single inversion circuit to be used for both directions of motion, again limiting mis-match. The output of the "one-over" circuit is amplified, and the polarity of the measured speed is restored. This analog voltage is proportional to retinal speed.

The measured retinal speed is subsequently added to the stored velocity. Figure 6 shows the schematic for the retinal velocity accumulation (positive feedback) and storage (analog

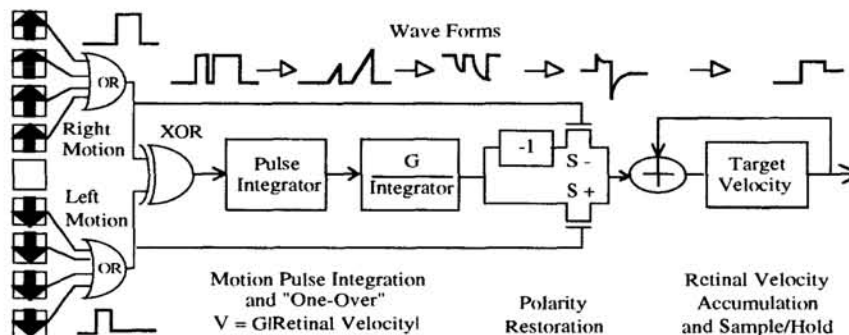

Figure 5: Architecture of the VLSI Smooth Pursuit System. Sketches of the wave forms for a fast leftward followed by a slow rightward retinal motion are shown.

memory). The output of the XOR gate in figure 5 is used by the sample-and-hold circuit to control sampling switches S1 and S2. During accumulation, the old stored velocity value, which is the current eye velocity, is isolated from the summed value. At the falling edge of the XOR output, the stored value on C2 is replaced by the new value on C1. This stored value is amplified using an off chip motor driver circuit, and used to move the chip. The gain of the motor driver can be finely controlled for optimal operation.

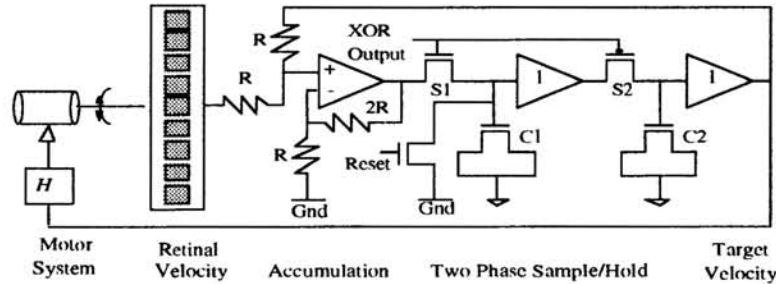

Figure 6: Schematic Retinal Velocity Error Accumulation, Storage and Motor Driver Systems.

Figure 7(a) shows a plot of one-over the measured integrated voltage as a function of on chip target speed. Due to noise in the integrator circuit, the dynamic range of the motion detection system is reduced to 2 orders of magnitude. However, the matching between left and right motion is unaffected by the integrator. The MOS "one-over" circuit, used to compute the analog reciprocal of the integrated voltage, exhibits only 0.06% deviation from a fitted line (Etienne-Cummings, 1993b). Figure 7(b) shows the measured increments in stored target velocity as a function of retinal (on-chip) speed. This is a test of all the circuit components of the tracking system. Linearity between retinal velocity increments and target velocity is observed, however matching between opposite motion has degraded. This is caused by the polarity restoration circuit since it is the only location where different circuits are used for opposite motion. On average, positive increments are a factor of 1.2 times larger than negative increments. The error bars shows the variation in velocity increments for different motion cells and different chips. The deviation is less than 15 %. The analog memory has a leakage of 10 mV/min and an asymmetric swing of 2 to -1 V, caused by the buffers. The dynamic range of the complete smooth pursuit system is measured to be 1.5 orders magnitude. The maximum speed of the system is adjustable by varying the integrator charging time. The maximum speed is ambient intensity dependent and ranges from 93 cm/s to 7 cm/s on-chip speed in

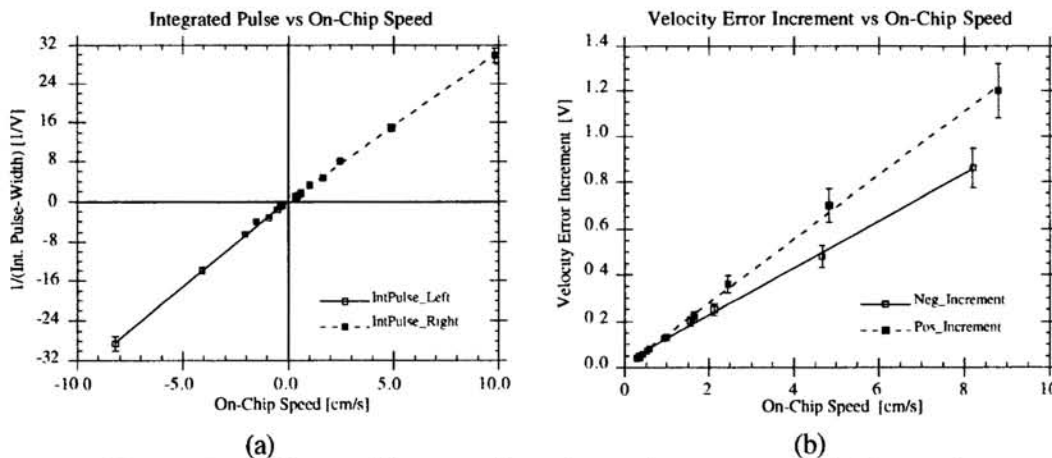

Figure 7. (a) Measured integrated motion pulse voltage. (b) Measured output for the complete smooth pursuit system.

bright ($250~\mu$W/cm$^2$) and dim ($2.5~\mu$W/cm$^2$) lighting, respectively. However, for any maximum speed chosen, the minimum speed is a factor of 0.03 slower. The minimum speed is limited by the discharge time of the temporal differentiators in the motion detection circuit to 0.004 cm/s on chip. The contrast sensitivity of this system proved to be the stumbling block, and it can not track objects in normal indoor lighting. However, all circuits components tested successfully when a light source is used as the target. Additional measured data can be found in (Etienne-Cummings, 1995). Further work will improve the contrast sensitivity, combat noise and also consider two dimensional implementations with target acquisition (saccades) capabilities.

## 4  CONCLUSION

A model for biological and silicon smooth pursuit has been presented. It combines the negative feed back and positive feedback models of Robinson and Wyatt and Pola. The smooth pursuit system is stable if the gain product of the retinal velocity detection system and the eye movement system is less than 2. VLSI implementation of this system has been performed and tested. The performance of the system suggests that wide range (92.9 - 0.004 cm/s retinal speed) target tracking is possible with a single chip focal plane system. To improve this chip's performance, care must be taken to limit noise, improve matching and increase contrast sensitivity. Future design should also include a saccadic component to re-capture escaped targets, similar to biological systems.

**References**

C. Bandera, "Foveal Machine Vision Systems", *Ph.D. Thesis*, SUNY Buffalo, New York, 1990

H. Barlow and W. Levick, "The Mechanism for Directional Selective Units in Rabbit's Retina", *Journal of Physiology*, Vol. 178, pp. 477-504, 1965

T. Delbruck, "Silicon Retina with Correlation-Based, Velocity-Tuned Pixels", *IEEE Transactions on Neural Networks*, Vol. 4:3, pp. 529-41, 1993

M. Eckert and G. Buchsbaum, "Effect of Tracking Strategies on the Velocity Structure of Two-Dimensional Image Sequences", J. Opt. Soc. Am., Vol. A10:7, pp. 1582-85, 1993

R. Etienne-Cummings *et al.*, "A New Temporal Domain Optical Flow Measurement Technique for Focal Plane VLSI Implementation", *Proceedings of CAMP 93*, M. Bayoumi, L. Davis and K. Valavanis (Eds.), pp. 241-251, 1993

R. Etienne-Cummings, R. Hathaway and J. Van der Spiegel, "An Accurate and Simple CMOS 'One-Over' Circuit", *Electronic Letters*, Vol. 29-18, pp. 1618-1620, 1993b

R. Etienne-Cummings *et al.*, "Real-Time Visual Target Tracking: Two Implementations of Velocity Based Smooth Pursuit", *Visual Information Processing IV*, SPIE Vol. 2488, Orlando, 17-18 April 1995

W. Reichardt, "Autocorrelation, A Principle for the Evaluation of Sensory Information by the Central Nervous System", *Sensory Communication*, Wiley, New York, 1961

D. Robinson, "The Mechanism of Human Smooth Pursuit Eye Movement", *Journal of Physiology ( London )* Vol. 180, pp. 569-591, 1965

M. Steinbach, "Pursuing the Perceptual Rather than the Retinal Stimuli", *Vision Research*, Vol. 16, pp. 1371-1376, 1976

H. Wyatt and J. Pola, "The Role of Perceived Motion in Smooth Pursuit Eye Movements", *Vision Research*, Vol. 19, pp. 613-618, 1979
